# Unsupervised Regression with Applications to Nonlinear System Identification

**Ali Rahimi**
Intel Research Seattle
Seattle, WA 98105
ali.rahimi@intel.com

**Ben Recht**
California Institute of Technology
Pasadena, CA 91125
brecht@ist.caltech.edu

## Abstract

We derive a cost functional for estimating the relationship between high-dimensional observations and the low-dimensional process that generated them with no input-output examples. Limiting our search to invertible observation functions confers numerous benefits, including a compact representation and no suboptimal local minima. Our approximation algorithms for optimizing this cost functional are fast and give diagnostic bounds on the quality of their solution. Our method can be viewed as a manifold learning algorithm that utilizes a prior on the low-dimensional manifold coordinates. The benefits of taking advantage of such priors in manifold learning and searching for the inverse observation functions in system identification are demonstrated empirically by learning to track moving targets from raw measurements in a sensor network setting and in an RFID tracking experiment.

## 1 Introduction

Measurements from sensor systems typically serve as a proxy for latent variables of interest. To recover these latent variables, the parameters of the sensor system must first be determined. When pairs of measurements and their corresponding latent variables are available, fully supervised regression techniques may be applied to learn a mapping between latent states and measurements. In many applications, however, latent states cannot be observed and only a diffuse prior on them is available. In such cases, marginalizing over the latent variables and searching for the model parameters using Expectation Maximization (EM) has become a popular approach [3, 9, 19]. Unfortunately, such algorithms are prone to local minima and require very careful initialization in practice.

Using a simple change-of-variable model, we derive an approximation algorithm for the *Unsupervised Regression* problem – estimating the nonlinear relationship between latent-states and their observations when no example pairs are available, when the observation function is invertible, and when the measurement noise is small. Our method is not susceptible to local minima and provides a guarantee on the quality of the recovered observation function. We identify conditions under which our estimate of the mapping is asymptotically consistent and empirically evaluate the quality of our solutions and their stability under variations of the prior. Because our algorithm takes advantage of an explicit prior on the latent variables, it recovers latent variables more accurately than manifold learning algorithms when applied to similar tasks.

Our method may be applied to estimate the observation function in nonlinear dynamical systems by enforcing a Markovian dynamics prior over the latent states. We demonstrate this approach to nonlinear system identification by learning to track a moving object in a field of completely uncalibrated sensor nodes whose measurement functions are unknown. Given that the object moves smoothly over time, our algorithm learns a function that maps the raw measurements from the sensor network to the target's location. In another experiment, we learn to track Radio Frequency ID (RFID)

tags given a sequence of voltage measurements induced by the tag in a set of antennae . Given only these measurements and that the tag moves smoothly over time, we can recover a mapping from the voltages to the position of the tag. These results are surprising because no parametric sensor model is available in either scenario. We are able to recover the measurement model up to an affine transform given only raw measurement sequences and a diffuse prior on the state sequence.

## 2   A diffeomorphic warping model for unsupervised regression

We assume that the set $\mathbf{X} = \{x_i\}_{1\ldots N}$ of latent variables is drawn (not necessarily iid) from a known distribution, $p_{\mathbf{X}}(\mathbf{X}) = p_{\mathbf{X}}(x_1, \cdots, x_N)$. The set of measurements $\mathbf{Y} = \{y_i\}_{1\ldots N}$ is the output of an unknown invertible nonlinearity applied to each latent variable, $y_i = f_0(x_i)$. We assume that observations, $y_i \in \mathcal{R}^D$, are higher dimensional than latent variables $x_i \in \mathcal{R}^d$. Computing a MAP estimate of $f_0$ requires marginalizing over $\mathbf{X}$ and maximizing over $f$. EM, or some other form of coordinate ascent on a Jensen bound of the likelihood, is a common way of estimating the parameters of this model, but such methods suffer from local minima.

Because we have assumed that $f_0$ is invertible and that there is no observation noise, this process describes a change of variables. The true distribution $p_{\mathbf{Y}}(\mathbf{Y})$ over $\mathbf{Y}$ can be computed in closed form using a generalization of the standard change of variables formula (see [14, thm 9.3.1] and [7, chap 11]):

$$p_{\mathbf{Y}}(\mathbf{Y}) = p_{\mathbf{Y}}(\mathbf{Y}; f_0) = p_{\mathbf{X}}(f_0^{-1}(y_1), \cdots, f_0^{-1}(y_N)) \prod_{i=1}^{N} \det \left( \nabla f \left( f_0^{-1}(y_i) \right)' \nabla f \left( f_0^{-1}(y_i) \right) \right)^{-\frac{1}{2}}. \tag{1}$$

The determinant corrects the warping of each infinitesimal volume element around $f_0^{-1}(y_i)$ by accounting for the stretching induced by the nonlinearity. The change of variables formula immediately yields a likelihood over $f$, circumventing the need for integrating over the latent variables.

We assume $f_0$ diffeomorphically maps a ball in $\mathcal{R}^d$ containing the data onto its image. In this case, there exists a function $g$ defined on an open set containing the image of $f$ such that $g(f(x)) = x$ and $\nabla g \nabla f = \mathbf{I}$ for all $x$ in the open set [5]. Consequently, we can substitute $g$ for $f^{-1}$ in (1) and, taking advantage of the identity $\det(\nabla f' \nabla f)^{-1} = \det \nabla g \nabla g'$, write its log likelihood as

$$l_{\mathbf{Y}}(\mathbf{Y}; g) = \log p_{\mathbf{Y}}(\mathbf{Y}; g) = \log p_{\mathbf{X}}(g(y_1), \ldots, g(y_N)) + \frac{1}{2} \sum_{i=1}^{N} \log \det \left( \nabla g(y_i) \nabla g(y_i)' \right). \tag{2}$$

For many common priors $p_{\mathbf{X}}$, the maximum likelihood $g$ yields an asymptotically consistent estimate of the true distribution $p_{\mathbf{Y}}$. When certain conditions on $p_{\mathbf{X}}$ are met (including stationarity, ergodicity, and $k$th-order Markov approximability), a generalized version of the Shannon-McMillan-Breiman theorem [1] guarantees that $\log p_{\mathbf{Y}}(\mathbf{Y}; g)$ asymptotically converges to the relative entropy rate between the true $p_{\mathbf{Y}}(\mathbf{Y})$ and $p_{\mathbf{Y}}(\mathbf{Y}; g)$. This quantity is maximized when these two distributions are equal. Therefore, if the true $p_{\mathbf{Y}}$ follows the change of variable model (1), the recovered $g$ converges to the true $f_0^{-1}$ in the sense that they both describe a change of variable from the prior distribution $p_{\mathbf{X}}$ to the distribution $p_{\mathbf{Y}}$.

Note that although our generative model assumes no observation noise, some noise in $\mathbf{Y}$ can be tolerated if we constrain our search over smooth functions $g$. This way, small perturbations in $y$ due to observation noise produce small perturbations in $g(y)$.

## 3   Approximation algorithms for finding the inverse mapping

We constrain our search for $g$ to a subset of smooth functions by requiring that $g$ have a finite representation as a weighted sum of positive definite kernels $k$ centered on observed data, $g(y) = \sum_{i=1}^{N} c_i k(y, y_i)$, with the weight vectors $c_i \in \mathcal{R}^d$. Accordingly, applying $g$ to the set of observations gives $g(\mathbf{Y}) = \mathbf{CK}$, where $\mathbf{C} = [c_1 \cdots c_N]$ and $\mathbf{K}$ is the kernel matrix with $K_{ij} = k(y_i, y_j)$. In addition, $\nabla g(y) = \mathbf{C} \Delta(y)$, where $\Delta(y)$ is an $N \times D$ matrix whose $i$th row is $\frac{\partial k(y_i, y)}{\partial y}$. We tune the smoothness of $g$ by regularizing (2) with the RKHS norm [17] of $g$. This norm has the form $\|g\|_k^2 = \operatorname{tr} \mathbf{CKC}'$, and the regularization parameter is set to $\frac{\lambda}{2}$.

For simplicity, we require $p_{\mathbf{X}}$ to be a Gaussian with mean zero and inverse covariance $\Omega_{\mathbf{X}}$, but we note our methods can be extended to any log-concave distribution. Substituting into (2) and adding the smoothness penalty on $g$, we obtain:

$$\max_{\mathbf{C}} \quad -\text{vec}\left(\mathbf{KC}'\right)'\Omega_{\mathbf{X}}\text{vec}\left(\mathbf{KC}'\right) - \lambda\text{tr}\mathbf{CKC}' + \sum_{i=1}^{N}\log\det\mathbf{C}\Delta(y_i)\Delta(y_i)'\mathbf{C}', \qquad (3)$$

where the $\text{vec}\left(\cdot\right)$ operator stacks up the columns of its matrix argument into a column vector.

Equation (3) is not concave in $\mathbf{C}$ and is likely to be hard to maximize exactly. This is because $\log\det(\mathbf{A}'\mathbf{A})$ is not concave for $\mathbf{A} \in \mathcal{R}^{d \times D}$. Since the cost is non-concave, gradient descent methods may converge to local minima. Such local minima, in addition to the burdensome time and storage requirements, rule out descent strategies for optimizing (3).

Our first algorithm for approximately solving this optimization problem constructs a semidefinite relaxation using a standard approach that replaces outer products of vectors with positive definite matrices. Rewrite (3) as

$$\max_{\mathbf{C}} \quad -\text{tr}\left(\mathbf{M}\text{vec}\left(\mathbf{C}'\right)\text{vec}\left(\mathbf{C}'\right)'\right) + \sum_{i=1}^{N}\log\det\left(\left[\text{tr}\,\mathbf{J}_i^{kl}\text{vec}\left(\mathbf{C}'\right)\text{vec}\left(\mathbf{C}'\right)'\right]\right) \qquad (4)$$

$$\mathbf{M} = (\mathbf{I}_d \otimes \mathbf{K})\Omega_{\mathbf{X}}(\mathbf{I}_d \otimes \mathbf{K}) + \lambda(\mathbf{I}_d \otimes \mathbf{K}) \quad , \qquad \mathbf{J}_i^{kl} = \mathbf{E}^{lk} \otimes \Delta(y_i)\Delta(y_i)' \qquad (5)$$

where the $kl$th entry of the matrix argument of the logdet is as specified, and the matrix $\mathbf{E}^{ij}$ is zero everywhere except for 1 in its $ij$th entry. This optimization is equivalent to

$$\max_{\mathbf{Z}\succeq 0} \quad -\text{tr}\left(\mathbf{MZ}\right) + \sum_{i=1}^{N}\log\det\left(\left[\text{tr}\,\mathbf{J}_i^{kl}\mathbf{Z}\right]\right), \qquad (6)$$

subject to the additional constraint that $\text{rank}(\mathbf{Z}) = 1$. Dropping the rank constraint yields a concave relaxation for (3). Standard interior point methods [20] or subgradient methods [2] can efficiently compute the optimal $\mathbf{Z}$ for this relaxed problem. A set of coefficients $\mathbf{C}$ can then be extracted from the top eigenvectors of the optimal $\mathbf{Z}$, yielding an approximate solution to (3). Since (6) without the rank constraint is a relaxation of (3), the optimum of (6) is an upper bound on that of (3). Thus we can bound the difference in the value of the extracted solution and that of the global maximum of (3). As we will see in the following section, this method produces high quality solutions for a diverse set of learning problems.

In practice, standard algorithms for (6) run slowly for large data sets, so we have developed an intuitive algorithm that also provides good approximations and runs much more quickly. The non-concave logdet term serves to prevent the optimal solution of (2) from collapsing to $g(y) = 0$, since $\mathbf{X} = 0$ is the most likely setting for the zero-mean Gaussian prior $p_{\mathbf{X}}$. To circumvent the non-concavity of the logdet term, we replace it with constraints requiring that the sample mean and covariance of $g(\mathbf{Y})$ match the expected mean and covariance of the random variables $\mathbf{X}$. These moment constraints prevent the optimal solution from collapsing to zero while remaining in the typical set of $p_{\mathbf{X}}$. The expected covariance of $\mathbf{X}$, denoted by $\bar{\Lambda}_{\mathbf{X}}$, can be computed by averaging the block diagonals of $\Omega_{\mathbf{X}}^{-1}$. However, the particular choice of $\bar{\Lambda}_{\mathbf{X}}$ only influences the final solution up to a scaling and rotation on $g$, so in practice, we set it to the identity matrix. We thus obtain the following optimization problem:

$$\min_{\mathbf{C}} \quad \text{vec}\left(\mathbf{KC}'\right)'\Omega_{\mathbf{X}}\text{vec}\left(\mathbf{KC}'\right) + \lambda\text{tr}\mathbf{CKC}' \qquad (7)$$

$$\text{s.t.} \quad \frac{1}{N}\mathbf{CK}(\mathbf{CK})' = \bar{\Lambda}_{\mathbf{X}} \qquad (8)$$

$$\frac{1}{N}\mathbf{CK1} = 0, \qquad (9)$$

where $\mathbf{1}$ is a column vector of 1s. This optimization problem searches for a $g$ that transforms observations into variables that are given high probability by $p_{\mathbf{X}}$ and match its stationary statistics. This is a quadratic minimization with a single quadratic constraint and, after eliminating the linear constraints with a change of variables, can be solved as a generalized eigenvalue problem [4].

# 4  Related Work

Manifold learning algorithms and unsupervised nonlinear system identification algorithms solve variants of the unsupervised regression problem considered here.

Our method provides a statistical model that augments manifold learning algorithms with a prior on latent variables. Our spectral algorithm from Section 3 reduces to a variant of KPCA [15] when $\mathbf{X}$ are drawn iid from a spherical Gaussian. By adopting a nearest-neighbors form for $g$ instead of the RBF form, we obtain an algorithm that is similar to embedding step of LLE [12, chap 5]. In addition to our use of dynamics, a notable difference between our method and principal manifold methods [16] is that instead of learning a mapping from states to observations, we learn mappings from observations to states. This reduces the storage and computational requirements when processing high-dimensional data. As far as we are aware, in the manifold learning literature, only Jenkins and Mataric [6] explicitly take temporal coherency into account, by increasing the affinity of temporally adjacent points and applying Isomap [18].

State-of-the-art nonlinear system identification techniques seek to recover all the parameters of a continuous hidden Markov chain with nonlinear state transitions and observation functions given noisy observations [3,8,9,19]. Because these models are so rich and have so many unknowns, these algorithms resort to coordinate ascent (for example, via EM), making them susceptible to local minima. In addition, each iteration of coordinate ascent requires some form of nonlinear smoothing over the latent variables, which is itself both computationally costly and becomes prone to local minima when the estimated observation function becomes non-invertible during the iterations. Further, because mappings from low-dimensional to high-dimensional vectors require many parameters to represent, existing approaches tend to be unsuitable for large-scale sensor network or image analysis problems. Our algorithms do not have local minima and represent the more compact inverse observation function where high-dimensional observations appear only in pairwise kernel evaluations.

Comparisons with a semi-supervised variant of these algorithms [13] show that weak priors on the latent variables are extremely informative and that additional labeled data is often only necessary to fix the coordinate system.

# 5  Experiments

The following experiments show that latent states and observation functions can be accurately and efficiently recovered up to a linear coordinate transformation given only raw measurements and a generic prior over the latent variables. We compare against various manifold learning and nonlinear system identification algorithms. We also show that our algorithm is robust to variations in the choice of the prior.

As a measure of quality, we report the *affine registration error*, the average residual per data point after registering the recovered latent variables with their ground truth values using an affine transformation: $err = \min_{\mathbf{A},b} \frac{1}{N} \sqrt{\sum_{t=1}^{N} \|\mathbf{A}x_t - x_t^0 + b\|_2^2}$, where $x_t^0$ is the ground truth setting for $x_t$. All of our experiments use a spherical Gaussian kernel. To define the Gaussian prior $p_{\mathbf{X}}$, we start with a linear Gaussian Markov chain $s_t = \mathbf{A}s_{t-1} + \omega_t$, where $\mathbf{A}$ and the covariance of $\omega$ are block diagonal and define $d$ Markov chains that evolve independently from each other according to Newtonian motion. The latent variables $x_t$ extract the position components of $s_t$. The inverse covariance matrix corresponding to this process can be obtained in closed form. More details and additional experiments can be found in [12].

We begin with a low-dimensional data set to simplify visualization and comparison to systems that do not scale well with the dimensionality of the observations. Figure 1(b) shows the embedding of a 1500 step 2D random walk shown in Figure 1(a) into $\mathcal{R}^3$ by the function $f(x,y) = (x, y\cos(2y), y\sin(2y))$. Note that the 2D walk was not generated by a linear Gaussian model, as it bounces off the edges of its bounding box. Lifted points were passed to our algorithm, which returned the 2D variables shown in Figure 1(c). The true 2D coordinates are recovered up to a scale, a flip, and some shrinking in the lower left corner. Therefore the recovered $g$ is close the inverse of the original mapping, up to a linear transform. Figure 1(d) shows states recovered by the algorithm of Roweis and Ghahramani [3]. Smoothing with the recovered function simply projects the

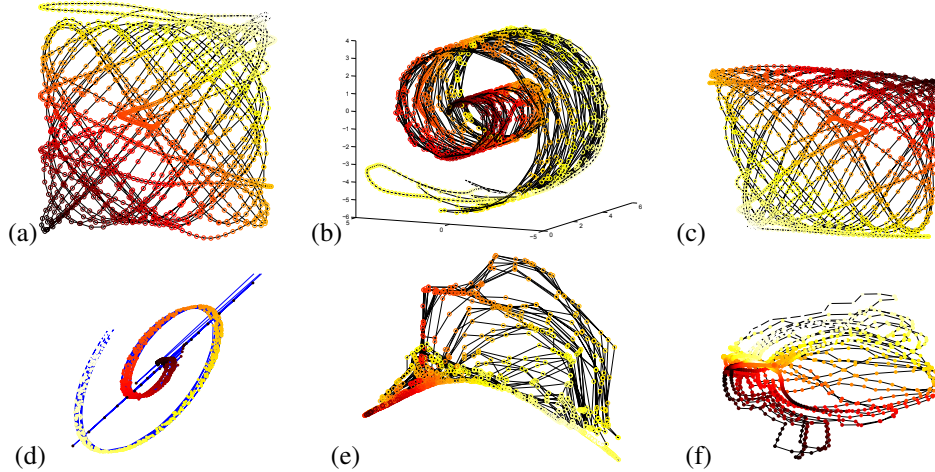

Figure 1: (a) 2D ground truth trajectory. Brighter colors indicate greater distance to the origin. (b) Embedding of the trajectory into $\mathcal{R}^3$. (c) Latent variables are recovered up to a linear transformation and minor distortion. Roweis-Ghahramani (d), Isomap (e), and Isomap+temporal coherence (f) recovered low-dimensional coordinates that exhibit folding and other artifacts that cannot be corrected by a linear transformation.

observations without unrolling the roll. The joint-max version of this algorithm took about an hour to converge on a 1Ghz Pentium III and converges only when started at solutions that are sufficiently close to the true solution. Our spectral algorithm took about 10 seconds. Isomap (Figure 1(e)) performs poorly on this data set due to the low sampling rate on the manifold and the fact that the true mapping $f$ is not isometric. Including temporal neighbors into Isomap's neighborhood structure (as per ST-Isomap) creates some folding, and the true underlying walk is not recovered (Figure 1(f)). KPCA (not shown) chooses a linear projection that simply eliminates the first coordinate. We found the optimal parameter settings for Isomap, KPCA, and ST-Isomap by a fine grid search over the parameter space of each algorithm.

The upper bound on the log-likelihood returned by the relaxation (6) serves as a diagnostic on the quality of our approximations. This bound was $-3.9 \times 10^{-3}$ for this experiment. Rounding the result of the relaxation returned a $g$ with log likelihood $-5.5 \times 10^{-3}$. The spectral approximation (7) also returned a solution with log likelihood $-5.5 \times 10^{-3}$, confirming our experience that these algorithms usually return similar solutions. For comparison, log-likelihood of KPCA's solution was $-1.69 \times 10^{-2}$, significantly less likely than our solutions, or the upper bound.

## 5.1   Learning to track in an uncalibrated sensor network

We consider an artificial distributed sensor network scenario where many sensor nodes are deployed randomly in a field in order to track a moving target (Figure 2(a)). The location of the sensor nodes is unknown, and the sensors are uncalibrated, so that it is not known how the position of the target maps to the reported measurements. This situation arises when it is not feasible to calibrate each sensor prior to deployment or when variations in environmental conditions affect each sensor differently. Given only the raw measurements produced by the network from watching a smoothly moving target, we wish to learn a mapping from these measurements to the location of the target, even though no functional form for the measurement model is available. A similar problem was considered by [11], who sought to recover the location of sensor nodes using off-the-shelf manifold learning algorithms.

Each latent state $x_t$ is the unknown position of the target at time $t$. The unknown function $f(x_t)$ gives the set of measurements $y_t$ reported by the sensor network at time $t$. Figure 2(b) shows the time series of measurements from observing the target. In this case, measurements were generated by having each sensor $s$ report its true distance $d_t^s$ to the target at time $t$ and passing it through a random nonlinearity of the form $\alpha^s \exp(-\beta^s d_t^s)$. Note that only $f$, not the measurement function of each sensor, needs be invertible. This is equivalent to requiring that a memoryless mapping from measurements to positions must exist.

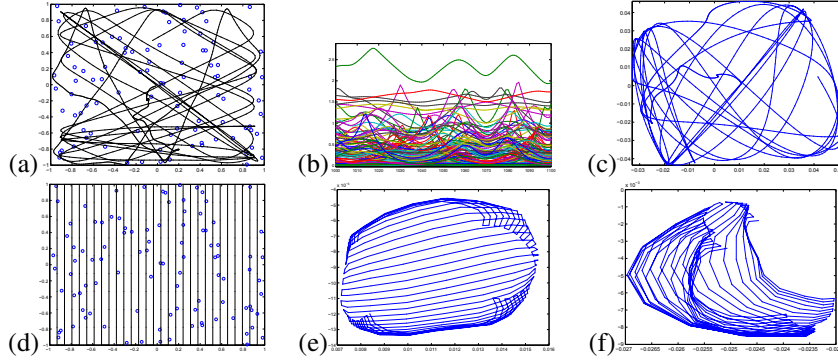

Figure 2: (a) A target followed a smooth trajectory (dotted line) in a field of 100 randomly placed uncalibrated sensors with random and unknown observation functions (circle). (b) Time series of measurements produced by the sensor network in response to the target's motion. (c) The recovered trajectory given only raw sensor measurements, and no information about the observation function (other than smoothness and invertibility). It is recovered up to scaling and a rotation. (d) To test the recovered mapping further, the target was made to follow a zigzag pattern. (e) Output of $g$ on the resulting measurements. The resulting trajectory is again similar to the ground truth zigzag, up to minor distortion. (f) The mapping obtained by KPCA cannot recover the zigzag, because KPCA does not utilize the prior on latent states.

Assuming only that the target vaguely follows linear-Gaussian dynamics, and given only the time series of the raw measurements from the sensor network, our learning algorithm finds a transformation that maps observations from the sensor network to the position of the target up to a linear coordinate transform (Figure 2(c)). The recovered function $g$ implicitly performs all the triangulation necessary for recovering the position of the target, even though the position or characteristics of the sensors were not known *a priori*. The bottom row of Figure 2 tests the recovered $g$ by applying it to a new measurement set. To show that this sensor network problem is not trivial, the figure also shows the output of the mapping obtained by KPCA.

## 5.2   Learning to Track with the *Sensetable*

The *Sensetable* is a hardware platform for tracking the position of radio frequency identification (RFID) tags. It consists of 10 antennae woven into a flat surface $30 \times 30$ cm. As an RFID tag moves along the flat surface, the strength of the RF signal induced by RFID tag in each antenna is reported, producing a time series of 10 numbers. We wish to learn a mapping from these 10 voltage measurements to the 2D position of the RFID tag. Previously, such a mapping was recovered by hand, by meticulous physical modeling of this system, followed by trial-and-error to refine these mappings; a process that took about 3 months in total [10]. We show that it is possible to recover this mapping automatically, up to an affine transformation, given only the raw time series of measurements generated by moving the RFID tag by hand on the *Sensetable* for about 5 minutes. This is a challenging task because the relationship between the tag's position and the observed measurements is highly oscillatory. (Figure 3(a)). Once it is learned, we can use the mapping to track RFID tags. This experiment serves as a real-world instantiation of the sensor network setup of the previous section in that each antenna effectively acts as an uncalibrated sensor node with an unknown and highly oscillatory measurement function.

Figure 3(b) shows the ground truth trajectory of the RFID tag in this data set. Given only the 5 minute-long time series of raw voltage measurements, our algorithm recovered the trajectory shown in Figure 3(c). These recovered coordinates are scaled down and flipped about both axes as compared to the ground truth coordinates. There is also some additional shrinkage in the upper right corner, but the coordinates are otherwise recovered accurately, with an affine registration error of 1.8 cm per pixel.

Figure 4 shows the result of LLE, KPCA, Isomap and ST-Isomap on this data set under their best parameter settings (again found by a grid search on each algorithm's search space). None of these algorithms recover low-dimensional coordinates that resemble the ground truth. LLE, in addition to collapsing the coordinates to one dimension, exhibits severe folding, obtaining an affine registration

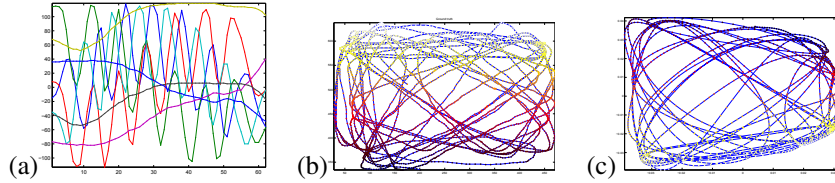

(a)　　　　　(b)　　　　　(c)

Figure 3: (a) The output of the *Sensetable* over a six second period, while moving the tag from the left edge of the table to the right edge. The observation function is highly complex and oscillatory. (b) The ground truth trajectory of the tag. Brighter points have greater ground truth $y$-value. (c) The trajectory recovered by our spectral algorithm is correct up to flips about both axes, a scale change, and some shrinkage along the edge.

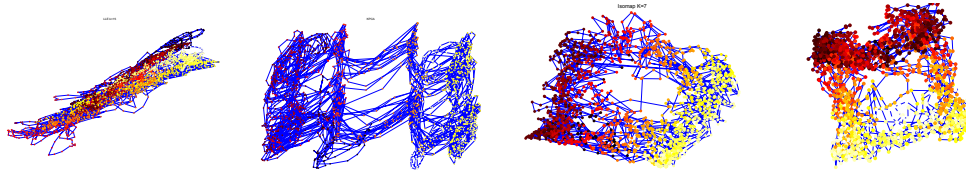

Figure 4: From left to right, the trajectories recovered by LLE, KPCA, Isomap, ST-Isomap. All of these trajectories exhibit folding and severe distortions.

error of 8.5 cm. KPCA also exhibited folding and large holes, with an affine registration error of 7.2 cm. Of these, Isomap performed the best with an affine registration error of 3.4 cm, though it exhibited some folding and a large hole in the center. Isomap with temporal coherency performed similarly, with a best affine registration error of 3.1 cm. Smoothing the output of these algorithms using the prior sometimes improves their accuracy by a few millimeters, but more often diminishes their accuracy by causing overshoots.

To further test the mapping recovered by our algorithm, we traced various trajectories with an RFID tag and passed the resulting voltages through the recovered $g$. Figure 5 plots the results (after a flip about the y-axis). These shapes resemble the trajectories we traced. Noise in the recovered coordinates is due to measurement noise.

The algorithm is robust to perturbations in $p_{\mathbf{X}}$. To demonstrate this, we generated 2000 random perturbations of the parameters of the inverse covariance of $\mathbf{X}$ used to generate the *Sensetable* results, and evaluated the resulting affine registration error. The random perturbations were generated by scaling the components of $\mathbf{A}$ and the diagonal elements of the covariance of $\omega$ over four orders of magnitude using a log uniform scaling. The affine registration error was below 3.6 cm for 38% of these 2000 perturbations. Typically, only the parameters of the kernel need to be tuned. In practice, we simply choose the kernel bandwidth parameter so that the minimum entry in $\mathbf{K}$ is approximately 0.1.

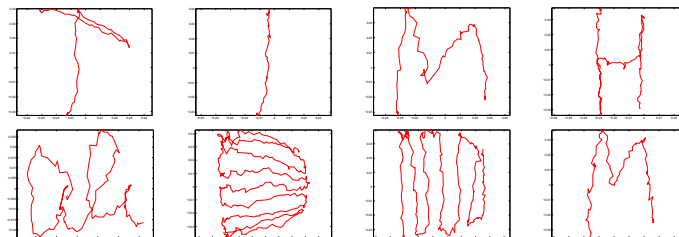

Figure 5: Tracking RFID tags using the recovered mapping.

# 6   Conclusions and Future Work

We have shown how to recover the latent variables in a dynamical system given an approximate prior on the dynamics of these variables and observations of these states through an unknown invertible nonlinearity. The requirement that the observation function be invertible is similar to the requirement in manifold learning algorithms that the manifold not intersect itself. Our algorithm enhances manifold learning algorithms by leveraging a prior on the latent variables. Because we search for a mapping from observations to unknown states (as opposed to from states to observations), we can devise algorithms that are stable and avoid local minima. We applied this methodology to learning to track objects given only raw measurements from sensors with no constraints on the observation model other than invertibility and smoothness.

We are currently evaluating various ways to relax the invertibility requirement on the observation function by allowing invertibility up to a linear subspace. We are also exploring different prior models, and experimenting with ways to jointly optimize over $g$ and the parameters of $p_{\mathbf{X}}$.

# References

[1] P.H. Algoet and T.M. Cover. A sandwich proof of the Shannon-McMillan-Breiman theorem. *The Annals of Probability*, 16:899–909, 1988.

[2] Aharon Ben-Tal and Arkadi Nemirovski. Non-euclidean restricted memory level method for large-scale convex optimization. *Mathematical Programming*, 102:407–456, 2005.

[3] Z. Ghahramani and S. Roweis. Learning nonlinear dynamical systems using an EM algorithm. In *Advances in Neural Information Processing Systems (NIPS)*, 1998.

[4] G. Golub and C.F. Van Loan. *Matrix Computations*. The Johns Hopkins University Press, 1989.

[5] V. Guilleman and A. Pollack. *Differential Topology*. Prentice Hall, Englewood Cliffs, New Jersey, 1974.

[6] O. Jenkins and M. Mataric. A spatio-temporal extension to isomap nonlinear dimension reduction. In *International Conference on Machine Learning (ICML)*, 2004.

[7] F. Jones. *Advanced Calculus*. http://www.owlnet.rice.edu/~fjones, unpublished.

[8] A. Juditsky, H. Hjalmarsson, A. Benveniste, B. Delyon, L. Ljung, J. Sjöberg, and Q. Zhang. Nonlinear black-box models in system identification: Mathematical foundations. *Automatica*, 31(12):1725–1750, 1995.

[9] N. D. Lawrence. Gaussian process latent variable models for visualisation of high dimensional data. In *Advances in Neural Information Processing Systems (NIPS)*, 2004.

[10] J. Patten, H. Ishii, J. Hines, and G. Pangaro. Sensetable: A wireless object tracking platform for tangible user interfaces. In *CHI*, 2001.

[11] N. Patwari and A. O. Hero. Manifold learning algorithms for localization in wireless sensor networks. In *International Conference on Acoustics, Speech, and Signal Processing (ICASSP)*, 2004.

[12] A. Rahimi. *Learning to Transform Time Series with a Few Examples*. PhD thesis, Massachusetts Institute of Technology, Computer Science and AI Lab, Cambridge, Massachusetts, USA, 2005.

[13] A. Rahimi, B. Recht, and T. Darrell. Learning appearance manifolds from video. In *Computer Vision and Pattern Recognition (CVPR)*, 2005.

[14] I. K. Rana. *An Introduction to Measure Theory and Integration*. AMA, second edition, 2002.

[15] B. Schölkopf, A. Smola, and K-R. Müller. Nonlinear component analysis as a kernel eigenvalue problem. *Neural Computation*, 10:1299–1319, 1998.

[16] A. Smola, S. Mika, B. Schoelkopf, and R. C. Williamson. Regularized principal manifolds. *Journal of Machine Learning*, 1:179–209, 2001.

[17] M. Pontil T. Evgeniou and T. Poggio. Regularization networks and support vector machines. *Advances in Computational Mathematics*, 2000.

[18] J. B. Tenenbaum, V. de Silva, and J. C. Langford. A global geometric framework for nonlinear dimensionality reduction. *Science*, 290(5500):2319–2323, 2000.

[19] H. Valpola and J. Karhunen. An unsupervised ensemble learning method for nonlinear dynamic state-space models. *Neural Computation*, 14(11):2647–2692, 2002.

[20] Lieven Vandenberghe, Stephen Boyd, and Shao-Po Wu. Determinant maximization with linear matrix inequality constraints. *SIAM Journal on Matrix Analysis and Applications*, 19(2):499–533, 1998.
